# Conditional Models of Identity Uncertainty with Application to Noun Coreference

**Andrew McCallum**[†]
[†]Department of Computer Science
University of Massachusetts Amherst
Amherst, MA 01003 USA
mccallum@cs.umass.edu

**Ben Wellner**[†*]
[*]The MITRE Corporation
202 Burlington Road
Bedford, MA 01730 USA
wellner@cs.umass.edu

## Abstract

Coreference analysis, also known as record linkage or identity uncertainty, is a difficult and important problem in natural language processing, databases, citation matching and many other tasks. This paper introduces several discriminative, conditional-probability models for coreference analysis, all examples of undirected graphical models. Unlike many historical approaches to coreference, the models presented here are relational—they do not assume that pairwise coreference decisions should be made independently from each other. Unlike other relational models of coreference that are generative, the conditional model here can incorporate a great variety of features of the input without having to be concerned about their dependencies—paralleling the advantages of conditional random fields over hidden Markov models. We present positive results on noun phrase coreference in two standard text data sets.

## 1   Introduction

In many domains—including computer vision, databases and natural language processing—we find multiple views, descriptions, or names for the same underlying object. Correctly resolving these references is a necessary precursor to further processing and understanding of the data. In computer vision, solving object correspondence is necessary for counting or tracking. In databases, performing record linkage or de-duplication creates a clean set of data that can be accurately mined. In natural language processing, coreference analysis finds the nouns, pronouns and phrases that refer to the same entity, enabling the extraction of relations among entities as well as more complex propositions.

Consider, for example, the text in a news article that discusses the entities *George Bush*, *Colin Powell*, and *Donald Rumsfeld*. The article contains multiple *mentions* of Colin Powell by different strings—"Secretary of State Colin Powell," "he," "Mr. Powell," "the Secretary"—and also refers to the other two entities with sometimes overlapping strings. The coreference task is to use the content and context of all the mentions to determine how many entities are in the article, and which mention corresponds to which entity.

This task is most frequently solved by examining individual pair-wise distance measures between mentions independently of each other. For example, database record-linkage and citation reference matching has been performed by learning a pairwise distance metric between records, and setting a distance threshold below which records are merged (Monge

& Elkan, 1997; McCallum et al., 2000; Bilenko & Mooney, 2002; Cohen & Richman, 2002). Coreference in NLP has also been performed with distance thresholds or pairwise classifiers (McCarthy & Lehnert, 1995; Ge et al., 1998; Soon et al., 2001; Ng & Cardie, 2002).

But these distance measures are inherently noisy and the answer to one pair-wise coreference decision may not be independent of another. For example, if we measure the distance between all of the three possible pairs among three mentions, two of the distances may be below threshold, but one above—an inconsistency due to noise and imperfect measurement. For example, "Mr. Powell" may be correctly coresolved with "Powell," but particular grammatical circumstances may make the model incorrectly believe that "Powell" is coreferent with a nearby occurrence of "she." Inconsistencies might be better resolved if the coreference decisions are made in *dependent relation* to each other, and in a way that accounts for the values of the multiple distances, instead of a threshold on single pairs independently.

Recently Pasula et al. (2003) have proposed a formal, relational approach to the problem of identity uncertainty using a type of Bayesian network called a Relational Probabilistic Model (Friedman et al., 1999). A great strength of this model is that it explicitly captures the dependence among multiple coreference decisions.

However, it is a generative model of the entities, mentions and all their features, and thus has difficulty using many features that are highly overlapping, non-independent, at varying levels of granularity, and with long-range dependencies. For example, we might wish to use features that capture the phrases, words and character n-grams in the mentions, the appearance of keywords anywhere in the document, the parse-tree of the current, preceding and following sentences, as well as 2-d layout information. To produce accurate generative probability distributions, the dependencies between these features should be captured in the model; but doing so can lead to extremely complex models in which parameter estimation is nearly impossible.

Similar issues arise in sequence modeling problems. In this area significant recent success has been achieved by replacing a generative model—hidden Markov models—with a conditional model—conditional random fields (CRFs) (Lafferty et al., 2001). CRFs have reduced part-of-speech tagging errors by 50% on out-of-vocabulary words in comparison with HMMs (Ibid.), matched champion noun phrase segmentation results (Sha & Pereira, 2003), and significantly improved extraction of named entities (McCallum & Li, 2003), citation data (Peng & McCallum, 2004), and the segmentation of tables in government reports (Pinto et al., 2003). Relational Markov networks (Taskar et al., 2002) are similar models, and have been shown to significantly improve classification of Web pages.

This paper introduces three conditional undirected graphical models for identity uncertainty. The models condition on the mentions, and generate the coreference decisions, (and in some cases also generate attributes of the entities). In the first most general model, the dependency structure is unrestricted, and the number of underlying entities explicitly appears in the model structure. The second and third models have no structural dependence on the number of entities, and fall into a class of Markov random fields in which inference corresponds to graph partitioning (Boykov et al., 1999).

After introducing the first two models as background generalizations, we show experimental results using the third, most specific model on a noun coreference problem in two different standard newswire text domains: broadcast news stories from the DARPA Automatic Content Extraction (ACE) program, and newswire articles from the MUC-6 corpus. In both domains we take advantage of the ability to use arbitrary, overlapping features of the input, including multiple grammatical features, string equality, substring, and acronym matches. Using the same features, in comparison with an alternative natural language processing technique, we reduce error by 33% and 28% in the two domains on proper nouns and by 10% on all nouns in the MUC-6 data.

## 2 Three Conditional Models of Identity Uncertainty

We now describe three possible configurations for conditional models of identity uncertainty, each progressively simpler and more specific than its predecessor. All three are based on conditionally-trained, undirected graphical models.

Undirected graphical models, also known as Markov networks or Markov random fields, are a type of probabilistic model that excels at capturing interdependent data in which causality among attributes is not apparent. We begin by introducing notation for mentions, entities and attributes of entities, then in the following subsections describe the likelihood, inference and estimation procedures for the specific undirected graphical models.

Let $\mathbf{E} = (E_1, ... E_m)$ be a collection of classes or "entities". Let $\mathbf{X} = (X_1, ... X_n)$ be a collection of random variables over observations or "mentions"; and let $\mathbf{Y} = (Y_1, ... Y_n)$ be a collection of random variables over integer identifiers, unique to each entity, specifying to which entity a mention refers. Thus the $y$'s are integers ranging from 1 to $m$, and if $Y_i = Y_j$, then mention $X_i$ is said to refer to the same underlying entity as $X_j$. For example, some particular entity $e_4$, *U.S. Secretary of State, Colin L. Powell*, may be mentioned multiple times in a news article that also contains mentions of other entities: $x_6$ may be "Colin Powell"; $x_9$ may be "he"; $x_{17}$ may be "the Secretary of State." In this case, the unique integer identifier for this entity, $e_4$, is 4, and $y_6 = y_9 = y_{17} = 4$.

Furthermore, entities may have attributes. Let $\mathbf{A}$ be a random variable over the collection of all attributes for all entities. Borrowing the notation of Relational Markov Networks (Taskar et al., 2002), we write the random variable over the attributes of entity $E_s$ as $E_s.\mathbf{A} = \{E_s.A_1, E_s.A_2, E_s.A_3, ...\}$. For example, these three attributes may be *gender, birth year*, and *surname*. Continuing the above example, then $e_4.a_1 = \mathsf{MALE}$, $e_4.a_2 = \mathsf{1937}$, and $e_4.a_3 = \mathsf{Powell}$. One can interpret the attributes as the values that should appear in the fields of a database record for the given entity. Attributes such as *surname* may take on one of the finite number of values that appear in the mentions of the data set.

We may examine many features of the mentions, $\mathbf{x}$, but since a conditional model doesn't generate them, we don't need random variable notation for them. Separate measured features of the mentions and entity-assignments, $\mathbf{y}$, are captured in different feature functions, $f(\cdot)$, over cliques in the graphical model. Although the functions may be real-valued, typically they are binary. The parameters of the model are associated with these different feature functions. Details and example feature functions and parameterizations are given for the three specific models below.

The task is then to find the most likely collection of entity-assignments, $\mathbf{y}$, (and optionally also the most likely entity attributes, $\mathbf{a}$), given a collection of mentions and their context, $\mathbf{x}$. A generative probabilistic model of identity uncertainty is trained to maximize $P(\mathbf{Y}, \mathbf{A}, \mathbf{X})$. A conditional probabilistic model of identity uncertainty is instead trained to maximize $P(\mathbf{Y}, \mathbf{A}|\mathbf{X})$, or simply $P(\mathbf{Y}|\mathbf{X})$.

### 2.1 Model 1: Groups of nodes for entities

First we consider an extremely general undirected graphical model in which there is a node for the mentions, $\mathbf{x}$,[1] a node for the entity-assignment of each mention, $y$, and a node for each of the attributes of each entity, $e.a$. These nodes are connected by edges in some unspecified structure, where an edge indicates that the values of the two connected random variables are dependent on each the other.

The parameters of the model are defined over cliques in this graph. Typically the parameters on many different cliques would be tied in patterns that reflect the nature of the repeated relational structure in the data. Patterns of tied parameters are common in many graphical models, including HMMs and other finite state machines (Lafferty et al., 2001), where they are tied across different positions in the input sequence, and by more complex patterns based on SQL-like queries, as in Markov Relational Networks (Taskar et al., 2002). Following the nomenclature of the later, these parameter-tying-patterns are called *clique templates*; each particular instance a template in the graph we call a *hit*.

For example, one clique template may specify a pattern consisting of two mentions, their entity-assignment nodes, and an entity's *surname* attribute node. The hits would consist of all possible combinations of such nodes. Multiple feature functions could then be run over each hit. One feature function might have value 1 if, for example, both mentions were assigned to the same entity as the surname node, and if the surname value appears as a substring in both mention strings (and value 0 otherwise).

The Hammersley-Clifford theorem stipulates that the probability of a particular set of values on the random variables in an undirected graphical model is a product of potential functions over cliques of the graph. Our cliques will be the hits, $\mathbf{h} = \{h, ...\}$, resulting from a set of clique templates, $\mathbf{t} = \{t, ...\}$. In typical fashion, we will write the probability distribution in exponential form, with each potential function calculated as a dot-product of feature functions, $f$, and learned parameters, $\lambda$,

$$P(\mathbf{y}, \mathbf{a}|\mathbf{x}) = \frac{1}{Z_{\mathbf{x}}} \exp \left( \sum_{t \in \mathbf{t}} \sum_{h_t \in \mathbf{h}_t} \sum_l \lambda_l f_l(\mathbf{y}, \mathbf{a}, \mathbf{x} : h_t) \right),$$

where $(\mathbf{y}, \mathbf{a}, \mathbf{x} : h_t)$ indicates the subset of the entity-assignment, attribute, and mention nodes selected by the clique template hit $h_t$; and $Z_{\mathbf{x}}$ is a normalizer to make the probabilities over all $\mathbf{y}$ sum to one (also known as the partition function).

The parameters, $\lambda$, can be learned by maximum likelihood from labeled training data. Calculating the partition function is problematic because there are a very large number of possible $\mathbf{y}$'s and $\mathbf{a}$'s. Loopy belief propagation or Gibbs sampling sampling have been used successfully in other similar situations, *e.g.* (Taskar et al., 2002).

However, note that both loopy belief propagation and Gibbs sampling only work over a graph with fixed structure. But in our problem the number of entities (and thus number of attribute nodes, and the domain of the entity-assignment nodes) is unknown. Inference in these models must determine for us the highest-probability number of entities.

In related work on a generative probabilistic model of identity uncertainty, Pasula et al. (2003), solve this problem by alternating rounds of Metropolis-Hastings sampling on a given model structure with rounds of Metropolis-Hastings to explore the space of new graph structures.

## 2.2 Model 2: Nodes for mention pairs, with attributes on mentions

To avoid the need to change the graphical model structure during inference, we now remove any parts of the graph that depend on the number of entities, $m$: (1) The per-mention entity-assignment nodes, $Y_i$, are random variables whose domain is over the integers 0 through $m$; we remove these nodes, replacing them with binary-valued random variables, $Y_{ij}$, over each pair of mentions, $(X_i, X_j)$ (indicating whether or not the two mentions are coreferent); although it is not strictly necessary, we also restrict the clique templates to operate over no more than two mentions (for efficiency). (2) The per-entity attribute nodes $A$ are removed and replaced with attribute nodes associated with each mention; we write $x_i.\mathbf{a}$ for the set of attributes on mention $x_i$.

Even though the clique templates are now restricted to pairs of mentions, this does not imply that pairwise coreference decisions are made independently of each other—they are

still highly dependent. Many pairs will overlap with each other, and constraints will flow through these overlaps. This point is reiterated with an example in the next subsection.

Notice, however, that it is possible for the model as thus far described to assign non-zero probability to an inconsistent set of entity-assignments, $\mathbf{y}$. For example, we may have an "inconsistent triangle" of coreference decisions in which $y_{ij}$ and $y_{jk}$ are 1, while $y_{ik}$ is 0. We can enforce the impossibility of all inconsistent configurations by adding inconsistency-checking functions $f_*(y_{ij}, y_{jk}, y_{ik})$ for all mention triples, with the corresponding $\lambda_*$'s fixed at negative infinity—thus assigning zero probability to them. (Note that this is simply a notational trick; in practice the inference implementation simply avoids any configurations of $\mathbf{y}$ that are inconsistent—a check that is simple to perform.) Thus we have

$$P(\mathbf{y}, \mathbf{a}|\mathbf{x}) = \frac{1}{Z_{\mathbf{x}}} \exp \left( \sum_{i,j,l} \lambda_l f_l(x_i, x_j, y_{ij}, x_i.\mathbf{a}, x_j.\mathbf{a}) + \sum_{i,j,k} \lambda_* f_*(y_{ij}, y_{jk}, y_{ik}) \right).$$

We can also enforce consistency among the attributes of coreferent mentions by similar means. There are many widely-used techniques for efficiently and drastically reducing the number of pair-wise comparisons, *e.g.* (Monge & Elkan, 1997; McCallum et al., 2000). In this case, we could also restrict $f_l(x_i, x_j, y_{ij}) \equiv 0, \forall y_{ij} = 0$.

### 2.3   Model 3: Nodes for mention pairs, graph partitioning with learned distance

When gathering attributes of entities is not necessary, we can avoid the extra complication of attributes by removing them from the model. What results is a straightforward, yet highly expressive, discriminatively-trained, undirected graphical model that can use rich feature sets and relational inference to solve identity uncertainty tasks. Determining the most likely number of entities falls naturally out of inference. The model is

$$P(\mathbf{y}|\mathbf{x}) = \frac{1}{Z_{\mathbf{x}}} \exp \left( \sum_{i,j,l} \lambda_l f_l(x_i, x_j, y_{ij}) + \sum_{i,j,k} \lambda_* f_*(y_{ij}, y_{jk}, y_{ik}) \right). \qquad (1)$$

Recently there has been increasing interest in study of the equivalence between graph partitioning algorithms and inference in certain kinds of undirected graphical models, *e.g.* (Boykov et al., 1999). This graphical model is an example of such a case. With some thought, one can straightforwardly see that finding the highest probability coreference solution, $\mathbf{y}^\star = \arg\max_{\mathbf{y}} P(\mathbf{y}|\mathbf{x})$, exactly corresponds to finding the graph partitioning of a (different) graph in which the mentions are the nodes and the edge weights are the (log) clique potentials on the pair of nodes $\langle x_i, x_j \rangle$ involved in their edge: $\sum_l \lambda_l f_l(x_i, x_j, y_{ij})$, where $f_l(x_i, x_j, 1) = -f_l(x_i, x_j, 0)$, and edge weights range from $-\infty$ to $+\infty$. Unlike classic mincut/maxflow binary partitioning, here the number of partitions (corresponding to entities) is unknown, but a single optimal number of partitions exists; negative edge weights encourage more partitions.

Graph partitioning with negative edge weights is NP-hard, but it has a history of good approximations, and several efficient algorithms to choose from. Our current experiments use an instantiation of the minimizing-disagreements Correlational Clustering algorithm in (Bansal et al., 2002). This approach is a simple yet effective partitioning scheme. It works by measuring the degree of inconsistency incurred by including a node in a partition, and making repairs. We refer the reader to Bansal et al. (2002) for further details.

The resulting solution does not make pairwise coreference decisions independently of each other. It has a significant "relational" nature because the assignment of a node to a partition (or, mention to an entity) depends not just on a single low distance measurement to one other node, but on its low distance measurement to all nodes in the partition (and furthermore on its high distance measurement to all nodes of all other partitions). For example, the "Mr. Powell"/"Powell"/"she" problem discussed in the introduction would be

prevented by this model because, although the distance between "Powell" and "she" might grammatically look low, the distance from "she" to another member of the same partition, ("Mr. Powell") is very high.

Interestingly, in our model, the distance measure between nodes is learned from labeled training data. That is, we use data, $\mathcal{D}$, in which the correct coreference partitions are known in order to learn a distance metric such that, when the same data is clustered, the correct partitions emerge. This is accomplished by maximum likelihood—adjusting the weights, $\lambda$, to maximize the product of Equation 1 over all instances $\langle \mathbf{x}, \mathbf{y} \rangle$ in the training set. Fortunately this objective function is concave—it has a single global maximum—and there are several applicable optimization methods to choose from, including gradient ascent, stochastic gradient ascent and conjugate gradient; all simply require the derivative of the objective function. The derivative of the log-likelihood, $L$, is

$$\frac{\partial L}{\partial \lambda_l} = \sum_{\langle \mathbf{x}, \mathbf{y} \rangle \in \mathcal{D}} \left( \sum_{i,j,l} f_l(x_i, x_j, y_{ij}) - \sum_{\mathbf{y}'} P_\Lambda(\mathbf{y}'|\mathbf{x}) \sum_{i,j,l} f_l(x_i, x_j, y'_{ij}) \right),$$

where $P_\Lambda(\mathbf{y}'|\mathbf{x})$ is defined by Equation 1, using the current set of $\lambda$ parameters, $\Lambda$, and $\sum_{\mathbf{y}'}$ is a sum over all possible partitionings.

The number of possible partitionings is exponential in the number of mentions, so for any reasonably-sized problem, we obviously must resort to approximate inference for the second expectation. A simple option is stochastic gradient ascent in the form of a voted perceptron (Collins, 2002). Here we calculate the gradient for a single training instance at a time, and rather than use a full expectation in the second line, simply using the single most likely (or nearly most likely) partitioning as found by a graph partitioning algorithm, and make progressively smaller steps in the direction of these gradients while cycling through the instances, $\langle \mathbf{x}, \mathbf{y} \rangle$ in the training data. Neither the full sum, $\sum_{\mathbf{y}'}$, or the partition function, $Z_\mathbf{x}$, need to be calculated in this case. Further details are given in (Collins, 2002).

## 3  Experiments with Noun Coreference

We present experimental results on natural language noun phrase coreference using Model 3 applied to two applicable data sets: the DARPA MUC-6 corpus, and a set of 117 stories from the broadcast news portion of the DARPA ACE data set. Both data sets have annotated coreferences. We pre-process both data sets with the Brill part-of-speech tagger.

We compare our Model 3 against two other techniques representing typical approaches to the problem of identity uncertainty. The first is single-link clustering with a threshold, (*single-link-threshold*), which is universally used in database record-linkage and citation reference matching (Monge & Elkan, 1997; Bilenko & Mooney, 2002; McCallum et al., 2000; Cohen & Richman, 2002). It forms partitions by simply collapsing the spanning trees of all mentions with pairwise distances below some threshold. For each experiment, the threshold was selected by cross validation.

The second technique, which we call *best-previous-match*, has been used in natural language processing applications (Morton, 1997; Ge et al., 1998; Ng & Cardie, 2002). It works by scanning linearly through a document, and associating each mention with its best-matching predecessor—best as measured with a single pairwise distance.

In our experiments, both single-link-threshold and best-previous-match implementations use a distance measure based on a binary maximum entropy classifier—matching the practice of Morton (1997) and Cohen and Richman (2002).

We use an identical feature set for all techniques, including our Method 3. The features, typical of those used in many other NLP coreference systems, are modeled after those in Ng and Cardie (2002). They include tests for string and substring matches, acronym

matches, parse-derived head-word matches, gender, WORDNET subsumption, sentence distance, distance in the parse tree; etc., and are detailed in an accompanying technical report. They are quite non-independent, and operate at multiple levels of granularity.

|  | ACE (Proper) | MUC-6 (Proper) | MUC-6 (All) |
|---|---|---|---|
| best-previous-match | 90.98 | 88.83 | 70.41 |
| single-link-threshold | 91.65 | 88.90 | 60.83 |
| Model 3 | 93.96 | 91.59 | 73.42 |

Table 1: F1 results on three data sets.

Table 1 shows standard MUC-style F1 scores for three experiments. In the first two experiments, we consider only proper nouns, and perform five-fold cross validation. In the third experiment, we perform the standard MUC evaluation, including all nouns—pronouns, common and proper—and use the standard 30/30 document train/test split; furthermore, as in Harabagiu et al. (2001), we consider only mentions that have a coreferent. Model 3 out-performs both the single-link-threshold and the best-previous-match techniques, reducing error by 28% over single-link-threshold on the ACE proper noun data, by 24% on the MUC-6 proper noun data, and by 10% over the best-previous-match technique on the full MUC-6 task. All differences from Model 3 are statistically significant. Historically, these data sets have been heavily studied, and even small gains have been celebrated.

Our overall results on MUC-6 are slightly better (with unknown statistical significance) than the best published results of which we are aware with a matching experimental design, Harabagiu et al. (2001), who reach 72.3% using the same training and test data.

## 4 Related Work and Conclusions

There has been much related work on identity uncertainty in various specific fields. Traditional work in de-duplication for databases or reference-matching for citations measures the distance between two records by some metric, and then collapses all records at a distance below a threshold, *e.g.* (Monge & Elkan, 1997; McCallum et al., 2000). This method is not relational, that is, it does not account for the inter-dependent relations among multiple decisions to collapse. Most recent work in the area has focused on learning the distance metric (Bilenko & Mooney, 2002; Cohen & Richman, 2002) not the clustering method.

Natural language processing has had similar emphasis and lack of emphasis respectively. Pairwise coreference learned distance measures have used decision trees (McCarthy & Lehnert, 1995; Ng & Cardie, 2002), SVMs (Zelenko et al., 2003), maximum entropy classifiers (Morton, 1997), and generative probabilistic models (Ge et al., 1998). But all use thresholds on a single pairwise distance, or the maximum of a single pairwise distance to determine if or where a coreferent merge should occur.

Pasula et al. (2003) introduce a generative probability model for identity uncertainty based on Probabilistic Relational Networks networks. Our work is an attempt to gain some of the same advantages that CRFs have over HMMs by creating conditional models of identity uncertainty. The models presented here, as instances of conditionally-trained undirected graphical models, are also instances of relational Markov networks (Taskar et al., 2002) and conditional Random fields (Lafferty et al., 2001). Taskar et al. (2002) briefly discuss clustering of dyadic data, such as people and their movie preferences, but not identity uncertainty or inference by graph partitioning.

Identity uncertainty is a significant problem in many fields. In natural language processing, it is not only especially difficult, but also extremely important, since improved coreference resolution is one of the chief barriers to effective data mining of text data. Natural language data is a domain that has particularly benefited from rich and overlapping feature representations—representations that lend themselves better to conditional probability models than generative ones (Lafferty et al., 2001; Collins, 2002; Morton, 1997). Hence our interest in conditional models of identity uncertainty.

## Acknowledgments

We thank Andrew Ng, Jon Kleinberg, David Karger, Avrim Blum and Fernando Pereira for helpful and insightful discussions. This work was supported in part by the Center for Intelligent Information Retrieval and in part by SPAWARSYSCEN-SD grant numbers N66001-99-1-8912 and N66001-02-1-8903, and DARPA under contract number F30602-01-2-0566 and in part by the National Science Foundation under NSF grant #IIS-0326249 and in part by the Defense Advanced Research Projects Agency (DARPA), through the Department of the Interior, NBC, Acquisition Services Division, under contract number NBCHD030010.

## Footnotes

[1] Even though there are many mentions in $\mathbf{x}$, because we are not generating them, we can represent them as a single node. This helps show that feature functions can ask arbitrary questions about various large and small subsets of the mentions and their context. We will still use $x_i$ to refer to the content and context of the $i$th mention.

## References

Bansal, N., Chawala, S., & Blum, A. (2002). Correlation clustering. *The 43rd Annual Symposium on Foundations of Computer Science (FOCS)* (pp. 238–247).

Bilenko, M., & Mooney, R. J. (2002). *Learning to combine trained distance metrics for duplicate detection in databases* (Technical Report Technical Report AI 02-296). Artificial Intelligence Laboratory, University of Texas at Austin, Austin, TX.

Boykov, Y., Veksler, O., & Zabih, R. (1999). Fast approximate energy minimization via graph cuts. *ICCV (1)* (pp. 377–384).

Cohen, W., & Richman, J. (2002). Learning to match and cluster entity names. *Proceedings of KDD-2002, 8th International Conference on Knowledge Discovery and Data Mining*.

Collins, M. (2002). Discriminative training methods for hidden markov models: Theory and experiments with perceptron algorithms.

Friedman, N., Getoor, L., Koller, D., & Pfeffer, A. (1999). Learning probabilistic relational models. *IJCAI* (pp. 1300–1309).

Ge, N., Hale, J., & Charniak, E. (1998). A statistical approach to anaphora resolution. *Proceedings of the Sixth Workshop on Very Large Corpora* (pp. 161–171).

Harabagiu, S., Bunescu, R., & Maiorano, S. (2001). Text and knowledge mining for coreference resolution. *Proceedings of the 2nd Meeting of the North American Chapter of the Association of Computational Linguistics (NAACL-2001)* (pp. 55–62).

Lafferty, J., McCallum, A., & Pereira, F. (2001). Conditional random fields: Probabilistic models for segmenting and labeling sequence data. *Proc. ICML* (pp. 282–289).

McCallum, A., & Li, W. (2003). Early results for named entity recognition with conditional random fields, feature induction and web-enhanced lexicons. *Seventh Conference on Natural Language Learning (CoNLL)*.

McCallum, A., Nigam, K., & Ungar, L. H. (2000). Efficient clustering of high-dimensional data sets with application to reference matching. *Knowledge Discovery and Data Mining* (pp. 169–178).

McCarthy, J. F., & Lehnert, W. G. (1995). Using decision trees for coreference resolution. *IJCAI* (pp. 1050–1055).

Monge, A. E., & Elkan, C. (1997). An efficient domain-independent algorithm for detecting approximately duplicate database records. *Research Issues on Data Mining and Knowledge Discovery*.

Morton, T. (1997). Coreference for NLP applications. *Proceedings ACL*.

Ng, V., & Cardie, C. (2002). Improving machine learning approaches to coreference resolution. *Fortieth Anniversary Meeting of the Association for Computational Linguistics (ACL-02)*.

Pasula, H., Marthi, B., Milch, B., Russell, S., & Shpitser, I. (2003). Identity uncertainty and citation matching. *Advances in Neural Information Processing (NIPS)*.

Peng, F., & McCallum, A. (2004). Accurate information extraction from research papers using conditional random fields. *Proceedings of Human Language Technology Conference and North American Chapter of the Association for Computational Linguistics (HLT-NAACL)*.

Pinto, D., McCallum, A., Lee, X., & Croft, W. B. (2003). Table extraction using conditional random fields. *Proceedings of the 26th ACM SIGIR*.

Sha, F., & Pereira, F. (2003). *Shallow parsing with conditional random fields* (Technical Report CIS TR MS-CIS-02-35). University of Pennsylvania.

Soon, W. M., Ng, H. T., & Lim, D. C. Y. (2001). A machine learning approach to coreference resolution of noun phrases. *Computational Linguistics*, *27*, 521–544.

Taskar, B., Abbeel, P., & Koller, D. (2002). Discriminative probabilistic models for relational data. *Eighteenth Conference on Uncertainty in Artificial Intelligence (UAI02)*.

Zelenko, D., Aone, C., & Richardella, A. (2003). Kernel methods for relation extraction. *Journal of Machine Learning Research (submitted)*.